# Spiking Boltzmann Machines

**Geoffrey E. Hinton**
Gatsby Computational Neuroscience Unit
University College London
London WC1N 3AR, UK
*hinton@gatsby.ucl.ac.uk*

**Andrew D. Brown**
Department of Computer Science
University of Toronto
Toronto, Canada
*andy@cs.utoronto.ca*

## Abstract

We first show how to represent sharp posterior probability distributions using real valued coefficients on broadly-tuned basis functions. Then we show how the precise times of spikes can be used to convey the real-valued coefficients on the basis functions quickly and accurately. Finally we describe a simple simulation in which spiking neurons learn to model an image sequence by fitting a dynamic generative model.

## 1 Population codes and energy landscapes

A perceived object is represented in the brain by the activities of many neurons, but there is no general consensus on how the activities of individual neurons combine to represent the multiple properties of an object. We start by focussing on the case of a single object that has multiple instantiation parameters such as position, velocity, size and orientation. We assume that each neuron has an ideal stimulus in the space of instantiation parameters and that its activation rate or probability of activation falls off monotonically in all directions as the actual stimulus departs from this ideal. The semantic problem is to define exactly what instantiation parameters are being represented when the activities of many such neurons are specified.

Hinton, Rumelhart and McClelland (1986) consider binary neurons with receptive fields that are convex in instantiation space. They assume that when an object is present it activates all of the neurons in whose receptive fields its instantiation parameters lie. Consequently, if it is known that only one object is present, the parameter values of the object must lie within the feasible region formed by the *intersection* of the receptive fields of the active neurons. This will be called a *conjunctive* distributed representation. Assuming that each receptive field occupies only a small fraction of the whole space, an interesting property of this type of "coarse coding" is that the bigger the receptive fields, the more accurate the representation. However, large receptive fields lead to a loss of resolution when several objects are present simultaneously.

When the sensory input is noisy, it is impossible to infer the exact parameters of objects so it makes sense for a perceptual system to represent the probability distribution across parameters rather than just a single best estimate or a feasible region. The full probability distribution is essential for correctly combining infor-

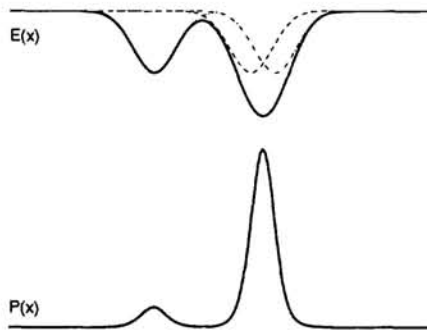

Figure 1: a) Energy landscape over a one-dimensional space. Each neuron adds a dimple (dotted line) to the energy landscape (solid line). b) The corresponding probability density. Where dimples overlap the corresponding probability density becomes sharper. Since the dimples decay to zero, the location of a sharp probability peak is not affected by distant dimples and multimodal distributions can be represented.

mation from different times or different sources. One obvious way to represent this distribution (Anderson and van Essen, 1994) is to allow each neuron to represent a fairly compact probability distribution over the space of instantiation parameters and to treat the activity levels of neurons as (unnormalized) mixing proportions. The semantics of this *disjunctive* distributed representation is precise, but the percepts it allows are not because it is impossible to represent distributions that are sharper than the individual receptive fields and, in high-dimensional spaces, the individual fields must be broad in order to cover the space. Disjunctive representations are used in Kohonen's self-organizing map which is why it is restricted to very low dimensional latent spaces.

The disjunctive model can be viewed as an attempt to approximate arbitrary smooth probability distributions by adding together probability distributions contributed by each active neuron. Coarse coding suggests a multiplicative approach in which the addition is done in the domain of energies (negative log probabilities). Each active neuron contributes an energy landscape over the whole space of instantiation parameters. The activity level of the neuron multiplies its energy landscape and the landscapes for all neurons in the population are added (Figure 1). If, for example, each neuron has a full covariance Gaussian tuning function, its energy landscape is a parabolic bowl whose curvature matrix is the inverse of the covariance matrix. The activity level of the neuron scales the inverse covariance matrix. If there are $k$ instantiation parameters then only $k + k(k + 1)/2$ real numbers are required to span the space of means and inverse covariance matrices. So the real-valued activities of $O(k^2)$ neurons are sufficient to represent arbitrary full covariance Gaussian distributions over the space of instantiation parameters.

Treating neural activities as multiplicative coefficients on additive contributions to energy landscapes has a number of advantages. Unlike disjunctive codes, vague distributions are represented by low activities so significant biochemical energy is only required when distributions are quite sharp. A central operation in Bayesian inference is to combine a prior term with a likelihood term or to combine two conditionally independent likelihood terms. This is trivially achieved by adding two energy landscapes[1].

## 2   Representing the coefficients on the basis functions

To perform perception at video rates, the probability distributions over instantiation parameters need to be represented at about 30 frames per second. This seems difficult using relatively slow spiking neurons because it requires the real-valued multiplicative coefficients on the basis functions to be communicated accurately and quickly using all-or-none spikes. The trick is to realise that when a spike arrives at another neuron it produces a postsynaptic potential that is a smooth function of time. So from the perspective of the postsynaptic neuron, the spike has been convolved with a smooth temporal function. By adding a number of these smooth functions together, with appropriate temporal offsets, it is possible to represent any smoothly varying sequence of coefficient values on a basis function, and this makes it possible to represent the temporal evolution of probability distributions as shown in Figure 2. The ability to vary the location of a spike in the single dimension of time thus allows real-valued control of the representation of probability distributions over multiple spatial dimensions.

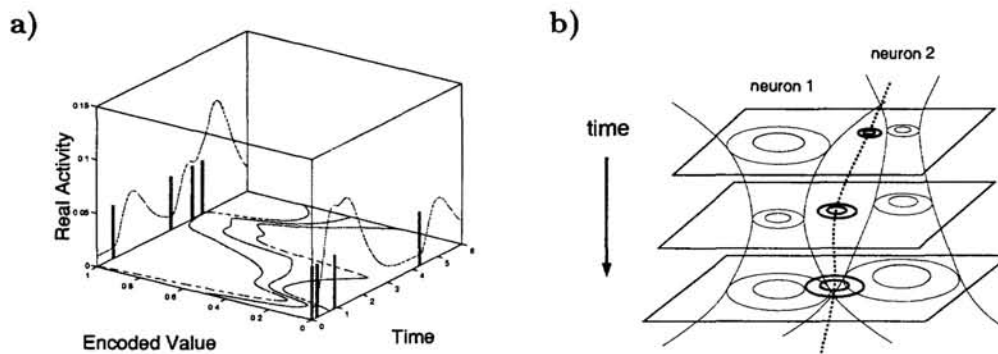

Figure 2: a)Two spiking neurons centered at 0 and 1 can represent the time-varying mean and standard deviation on a single spatial dimension. The spikes are first convolved with a temporal kernel and the resulting activity values are treated as exponents on Gaussian distributions centered at 0 and 1. The ratio of the activity values determines the mean and the sum of the activity values determines the inverse variance. b) The same method can be used for two (or more) spatial dimensions. Time flows from top to bottom. Each spike makes a contribution to the energy landscape that resembles an hourglass (thin lines). The waist of the hourglass corresponds to the time at which the spike has its strongest effect on some post-synaptic population. By moving the hourglasses in time, it is possible to get whatever temporal cross-sections are desired (thick lines) provided the temporal sampling rate is comparable to the time course of the effect of a spike.

Our proposed use of spike timing to convey real values quickly and accurately does not require precise coincidence detection, sub-threshold oscillations, modifiable time delays, or any of the other paraphernalia that has been invoked to explain how the brain could make effective use of the single, real-valued degree of freedom in the timing of a spike (Hopfield, 1995).

The coding scheme we have proposed would be far more convincing if we could show how it was learned and could demonstrate that it was effective in a simulation. There are two ways to design a learning algorithm for such spiking neurons. We could work in the relatively low-dimensional space of the instantiation parameters and design the learning to produce the right representations and interactions between representations in this space. Or we could treat this space as an implicit emergent property of the network and design the learning algorithm to optimize

some objective function in the much higher-dimensional space of neural activities in the hope that this will create representations that can be understood using the implicit space of instantiation parameters. We chose the latter approach.

# 3 A learning algorithm for restricted Boltzmann machines

Hinton (1999) describes a learning algorithm for probabilistic generative models that are composed of a number of experts. Each expert specifies a probability distribution over the visible variables and the experts are combined by multiplying these distributions together and renormalizing.

$$p(\mathbf{d}|\theta_1...\theta_n) = \frac{\Pi_m p_m(\mathbf{d}|\theta_m)}{\sum_i \Pi_m p_m(\mathbf{c}_i|\theta_m)} \qquad (1)$$

where $\mathbf{d}$ is a data vector in a discrete space, $\theta_m$ is all the parameters of individual model $m$, $p_m(\mathbf{d}|\theta_m)$ is the probability of $\mathbf{d}$ under model $m$, and $i$ is an index over all possible vectors in the data space.

The coding scheme we have described is just a product of experts in which each spike is an expert. We first summarize the Product of Experts learning rule for a restricted Boltzmann machine (RBM) which consists of a layer of stochastic binary visible units connected to a layer of stochastic binary hidden units with no intralayer connections. We then extend RBM's to deal with temporal data.

In an RBM, each hidden unit is an expert. When it is off it specifies a uniform distribution over the states of the visible units. When it is on, its weight to each visible unit specifies the log odds that the visible unit is on. Multiplying together the distributions specified by different hidden units is achieved by adding the log odds. Inference in an RBM is much easier than in a causal belief net because there is no explaining away. The hidden states, $s_j$, are conditionally independent given the visible states, $s_i$, and the distribution of $s_j$ is given by the standard logistic function $\sigma$: $p(s_j = 1) = \sigma(\sum_i w_{ij} s_i)$. Conversely, the hidden states of an RBM are *marginally* dependent so it is easy for an RBM to learn population codes in which units may be highly correlated. It is hard to do this in causal belief nets with one hidden layer because the generative model of a causal belief net assumes marginal independence.

An RBM can be trained by following the gradient of the log likelihood of the data:

$$\Delta w_{ij} = \epsilon \left( < s_i s_j >^0 - < s_i s_j >^\infty \right) \qquad (2)$$

where $< s_i s_j >^0$ is the expected value of $s_i s_j$ when data is clamped on the visible units and the hidden states are sampled from their conditional distribution given the data, and $< s_i s_j >^\infty$ is the expected value of $s_i s_j$ after prolonged Gibbs sampling that alternates between sampling from the conditional distribution of the hidden states given the visible states and vice versa.

This learning rule not work well because the sampling noise in the estimate of $< s_i s_j >^\infty$ swamps the gradient. It is far more effective to maximize the *difference* between the log likelihood of the data and the log likelihood of the one-step reconstructions of the data that are produced by first picking binary hidden states from their conditional distribution given the data and then picking binary visible states from their conditional distribution given the hidden states. The gradient of the log

likelihood of the one-step reconstructions is complicated because changing a weight changes the probability distribution of the reconstructions:

$$\frac{\partial L^1}{\partial w_{ij}} = <s_i s_j>^1 - <s_i s_j>^\infty + \frac{\partial Q^1}{\partial w_{ij}} \times \frac{\partial L^1}{\partial Q^1} \qquad (3)$$

where $Q^1$ is the distribution of the one-step reconstructions of the training data and $Q^\infty$ is the equilibrium distribution (*i.e.* the stationary distribution of prolonged Gibbs sampling). Fortunately, the cumbersome third term is sufficiently small that ignoring it does not prevent the vector of weight changes from having a positive cosine with the true gradient of the difference of the log likelihoods so the following very simple learning rule works much better than Eq. 2.

$$\Delta w_{ij} = \epsilon \left( <s_i s_j>^0 - <s_i s_j>^1 \right) \qquad (4)$$

## 4   Restricted Boltzmann machines through time

Using a restricted Boltzmann machine we can represent time by *spatializing* it, *i.e.* taking each visible unit, $i$, and hidden unit, $j$, and replicating them through time with the constraint that the weight $w_{ij\tau}$ between replica $t$ of $i$ and replica $t + \tau$ of $j$ does not depend on $t$. To implement the desired temporal smoothing, we also force the weights to be a smooth function of $\tau$ that has the shape of the temporal kernel, shown in Figure 3. The only remaining degree of freedom in the weights between replicas of $i$ and replicas of $j$ is the scale of the temporal kernel and it is this scale that is learned. The replicas of the visible and hidden units still form a bipartite graph and the probability distribution over the hidden replicas can be inferred exactly without considering data that lies further into the future than the width of the temporal kernel.

One problem with the restricted Boltzmann machine when we spatialize time is that hidden units at one time step have no memory of their states at previous time steps; they only see the data. If we were to add undirected connections between hidden units at different time steps, then the architecture would return to a fully connected Boltzmann machine in which the hidden units are no longer conditionally independent given the data. A useful trick borrowed from Elman nets is to allow the hidden units to see their previous states, but to treat these observations like data that cannot be modified by future hidden states. Thus, the hidden states may still be inferred independently without resorting to Gibbs sampling. The connections between hidden layer weights also follow the time course of the temporal kernel. These connections act as a predictive prior over the hidden units. It is important to note that these forward connections are not required for the network to model a sequence, but only for the purposes of extrapolating into the future.

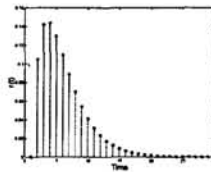

Figure 3: The form of the temporal kernel.

Now the probability that $s_j(t) = 1$ given the states of the visible units is,

$$P(s_j(t) = 1) = \sigma \left( \sum_i w_{ij} h_i(t) + \sum_k w_{kj} h_k(t) \right).$$

where $h_i(t)$ is the convolution of the history of visible unit $i$ with the temporal kernel,

$$h_i(t) = \sum_{\tau=0}^{\infty} s_i(t - \tau) r(\tau),$$

and $h_k(t)$, the convolution of the hidden unit history, is computed similarly. [2] Learning the weights follows immediately from this formula for doing inference. In the positive phase the visible units are clamped at each time step and the posterior of the hidden units conditioned on the data is computed (we assume zero boundary conditions for time before $t = 0$). Then in the negative phase we sample from the posterior of the hidden units, and compute the distribution over the visible units at each time step given these hidden unit states. In each phase the correlations between the hidden and visible units are computed and the learning rule is,

$$\Delta w_{ij} = \sum_{t=0}^{\infty} \sum_{\tau=0}^{\infty} r(\tau) \left( \langle s_j(t) s_i(t - \tau) \rangle^0 - \langle s_j(t) s_i(t - \tau) \rangle^1 \right).$$

## 5   Results

We trained this network on a sequence of 8x8 synthetic images of a Gaussian blob moving in a circular path. In the following diagrams we display the time sequence of images as a matrix. Each row of the matrix represents a single image with its pixels stretched out into a vector in scanline order, and each column is the time course of a single pixel. The intensity f the pixel is represented by the area of the white patch. We used 20 hidden units. Figure 5a shows a segment (200 time steps) of the time series which was used in training. In this sequence the period of the blob is 80 time steps.

Figure 5b shows how the trained model reconstructs the data after we sample from the hidden layer units. Once we have trained the model it is possible to do forecasting by clamping visible layer units for a segment of a sequence and then doing iterative Gibbs sampling to generate future points in the sequence. Figure 5c shows that given 50 time steps from the series, the model can predict reasonably far into the future, before the pattern dies out. One problem with these simulations is that we are treating the real valued intensities in the images as probabilities. While this works for the blob images, where the values can be viewed as the probabilities of pixels in a binary image being on, this is not true for more natural images.

## 6   Discussion

In our initial simulations we used a causal sigmoid belief network (SBN) rather than a restricted Boltzmann machine. Inference in an SBN is *much* more difficult than in an RBM. It requires Gibbs sampling or severe approximations, and even if a temporal kernel is used to ensure that a replica of a hidden unit at one time

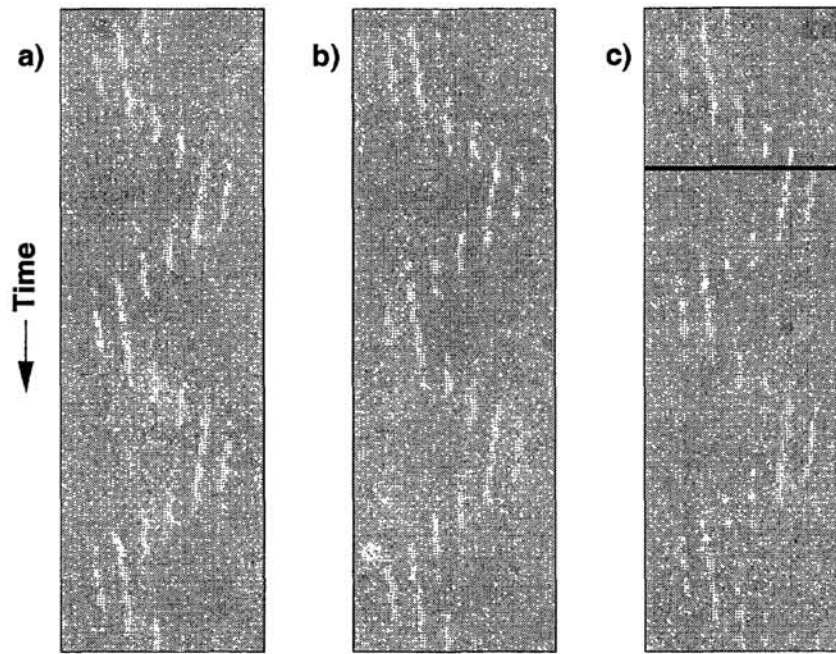

Figure 4: a) The original data, b) reconstruction of the data, and c) prediction of the data given 50 time steps of the sequence. The black line indicates where the prediction begins.

has no connections to replicas of visible units at very different times, the posterior distribution of the hidden units still depends on data far in the future. The Gibbs sampling made our SBN simulations very slow and the sampling noise made the learning far less effective than in the RBM. Although the RBM simulations seem closer to biological plausibility, they too suffer from a major problem. To apply the learning procedure it is necessary to reconstruct the data from the hidden states and we do not know how to do this without interfering with the incoming datastream. In our simulations we simply ignored this problem by allowing a visible unit to have both an observed value and a reconstructed value at the same time.

## Acknowledgements

We thank Zoubin Ghahramani, Peter Dayan, Rich Zemel, Terry Sejnowski and Radford Neal for helpful discussions. This research was funded by grants from the Gatsby Foundation and NSERC.

## Footnotes

[1]We thank Zoubin Ghahramani for pointing out that another important operation, convolving a probability distribution with Gaussian noise, is a difficult non-linear operation on the energy landscape.

[2]Computing the conditional probability distribution over the visible units given the hidden states is done in a similar fashion, with the caveat that the weights in each direction must be symmetric. Thus, the convolution is done using the reverse kernel.

## References

Anderson, C.H. & van Essen, D.C (1994). Neurobiological computational systems. In J.M Zureda, R.J. Marks, & C.J. Robinson (Eds.), *Computational Intelligence Imitating Life* 213-222. New York: IEEE Press.

Hinton, G. E. (1999) Products of Experts. *ICANN 99: Ninth international conference on Artificial Neural Networks*, Edinburgh, 1-6.

Hinton, G. E., McClelland, J. L., & Rumelhart, D. E. (1986) Distributed representations. In Rumelhart, D. E. and McClelland, J. L., editors, *Parallel Distributed Processing: Explorations in the Microstructure of Cognition. Volume 1: Foundations*, MIT Press, Cambridge, MA.

Hopfield, J. (1995). Pattern recognition computation using action potential timing for stimulus representation. *Nature,* **376**, 33-36.